# A Framework for Non-rigid Matching and Correspondence

**Suguna Pappu, Steven Gold, and Anand Rangarajan**[1]
Departments of Diagnostic Radiology and Computer Science
and the Yale Neuroengineering and Neuroscience Center
Yale University New Haven, CT 06520-8285

## Abstract

Matching feature point sets lies at the core of many approaches to object recognition. We present a framework for non-rigid matching that begins with a skeleton module, affine point matching, and then integrates multiple features to improve correspondence and develops an object representation based on spatial regions to model local transformations. The algorithm for feature matching iteratively updates the transformation parameters and the correspondence solution, each in turn. The affine mapping is solved in closed form, which permits its use for data of any dimension. The correspondence is set via a method for two-way constraint satisfaction, called *softassign*, which has recently emerged from the neural network/statistical physics realm. The complexity of the non-rigid matching algorithm with multiple features is the same as that of the affine point matching algorithm. Results for synthetic and real world data are provided for point sets in 2D and 3D, and for 2D data with multiple types of features and parts.

## 1 Introduction

A basic problem of object recognition is that of matching– how to associate sensory data with the representation of a known object. This entails finding a transformation that maps the features of the object model onto the image, while establishing a correspondence between the spatial features. However, a tractable class of transformation, e.g., affine, may not be sufficient if the object is non-rigid or has relatively independent parts. If there is noise or occlusion, spatial information alone may not be adequate to determine the correct correspondence. In our previous work in spatial point matching [1], the 2D affine transformation was decomposed into its

physical component elements, which does not generalize easily to 3D, and so, only a rigid 3D transformation was considered.

We present a framework for non-rigid matching that begins with solving the basic affine point matching problem. The algorithm iteratively updates the affine parameters and correspondence in turn, each as a function of the other. The affine transformation is solved in closed form, which lends tremendous flexibility– the formulation can be used in 2D or 3D. The correspondence is solved by using a *softassign* [1] procedure, in which the two-way assignment constraints are solved without penalty functions. The accuracy of the correspondence is improved by the integration of multiple features. A method for non-rigid parameter estimation is developed, based on the assumption of a well-articulated model with distinct regions, each of which may move in an affine fashion, or can be approximated as such. Umeyama [3] has done work on parameterized parts using an exponential time tree search technique, and Wakahara [4] on local affine transforms, but neither integrates multiple features nor explicitly considers the non-rigid matching case, while expressing a one-to-one correspondence between points.

## 2 Affine Point Matching

The affine point matching problem is formulated as an optimization problem for determining the correspondence and affine transformation between feature points. Given two sets of data points $\hat{X}_j \in R^{n-1}$, $n = 3, 4 \ldots$, $j = 1, \ldots, J$, the image, and $\hat{Y}_k \in R^{n-1}$, $n = 3, 4, \ldots, k = 1, \ldots, K$, the model, find the correspondence and associated affine transformation that best maps a subset of the image points onto a subset of the model point set. These point sets are expressed in homogeneous coordinates, $X_j = (1, \hat{X}_j)$, $Y_k = (1, \hat{Y}_k)$. $\{a_{ij}\} = A \in R^{n \times n}$ is the affine transformation matrix. Note that $\{a_{1j} = 0 \; \forall j\}$ because of the homogeneous coordinates. Define the *match* variable $M_{jk}$ where $M_{jk} \in [0, 1]$. For a given match matrix $\{M_{jk}\}$, transformation $A$ and $I$, an identity matrix of dimension $n$, $\sum_{j,k} M_{jk} \|X_j - (A + I)Y_k\|^2$ expresses the similarity between the point sets. The term $-\alpha \sum_{j,k} M_{jk}$, with parameter $\alpha > 0$ is appended to this to encourage matches (else $M_{jk} = 0 \; \forall \; j, \; k$ minimizes the function). To limit the range of transformations, the terms of the affine matrix are regularized via a term $\lambda tr(A^T A)$ in the objective function, with parameter $\lambda$, where $tr(.)$ denotes the trace of the matrix. Physically, $X_j$ may fully match to one $Y_k$, partially match to several, or may not match to any point. A similar constraint holds for $Y_k$. These are expressed as the constraints in the following optimization problem:

$$\min_{A,M} \sum_{j,k} M_{jk}\|X_j - (A+I)Y_k\|^2 + \lambda tr(A^T A) - \alpha \sum_{j,k} M_{jk} \qquad (1)$$

$$\text{s.t.} \quad \sum_j M_{jk} \leq 1, \; \forall k, \; \sum_k M_{jk} \leq 1, \; \forall j \text{ and } M_{jk} \geq 0$$

To begin, slack variables $M_{j,K+1}$ and $M_{J+1,k}$ are introduced so that the inequality constraints can be transformed into equality constraints: $\sum_{j=1}^{J+1} M_{jk} = 1$, $\forall k$ and $\sum_{k=1}^{K+1} M_{jk} = 1$, $\forall j$. $M_{j,K+1} = 1$ indicates that $X_j$ does not match to any point in $Y_k$. An equivalent unconstrained optimization problem to (2) is derived by relaxing the constraints via Lagrange parameters $\mu_j$, $\nu_k$, and introducing an $x \log x$ barrier function, indexed by a parameter $\beta$. A similar technique was used

[2] to solve the assignment problem. The energy function used is:

$$\min_{A,M} \max_{\mu,\nu} \sum_{j,k} M_{jk} \|X_j - (A+I)Y_k\|^2 + \lambda tr(A^T A) - \alpha \sum_{j,k} M_{jk} + \sum_{j}^{J} \mu_j \left( \sum_{k=1}^{K+1} M_{jk} - 1 \right)$$

$$+ \sum_{k}^{K} \nu_k \left( \sum_{j=1}^{J+1} M_{jk} - 1 \right) + \frac{1}{\beta} \sum_{j=1}^{J+1} \sum_{k=1}^{K+1} M_{jk} (\log M_{jk} - 1)$$

This is to be minimized with respect to the match variables and affine parameters while satisfying the constraints via Lagrange parameters. Using the recently developed *softassign* technique, we satisfy the constraints explicitly. When $A$ is fixed, we have an assignment problem. Following the development in [1], the assignment constraints are satisfied using softassign, a technique for satisfying two-way (assignment) constraints without a penalty term that is analogous to softmax which enforces a one-way constraint. First, the match variables are initialized:

$$M_{jk}^*(A) = \exp(-\beta \|X_j - (I+A)Y_k\|^2 - \alpha) \tag{2}$$

This is followed by repeated row-column normalization of the match variables until a stopping criterion is reached:

$$M_{jk} = \frac{M_{jk}}{\sum_{j'} M_{j'k}} \text{ then } M_{jk} = \frac{M_{jk}}{\sum_{k'} M_{jk'}} \tag{3}$$

When the correspondence between the two point sets is fixed, $A$ can be solved in closed form, by holding $M$ fixed in the objective function, and differentiating and solving for $A$:

$$A = A^*(M) = \left( \sum_{j,k} M_{jk} (X_j Y_k^T - Y_k Y_k^T) \right) \left( \sum_{j,k} M_{jk} Y_k Y_k^T + \lambda I \right)^{-1} \tag{4}$$

The algorithm is summarized as:

1. INITIALIZE: Variables: $A = 0$, $M = 0$
   Parameters: $\beta_{\text{initial}}, \beta_{\text{update}}, \beta_{\text{final}}$ $T = $ Inner loop iterations, $\lambda$
2. ITERATE: Do $T$ times for a fixed value of $\beta$
   Softassign: Re-initialize $M^*(A)$ and then (Eq. 2) until $\Delta M$ small
   $A^*(M)$ updated (Eq. 4)
3. UPDATE: While $\beta < \beta_{\text{final}}$, $\beta \leftarrow \beta * \beta_{\text{update}}$, Return to 2.

The complexity of the algorithm is $O(JK)$. Starting with small $\beta_{\text{initial}}$ permits many partial correspondences in the initial solution for $M$. As $\beta$ increases the correspondence becomes more refined. For large $\beta_{\text{final}}$, $M$ approaches a permutation matrix (adjusting appropriately for the slack variables).

## 3 Nonrigid Feature Matching: Affine Quilts

Recognition of an object requires many different types of information working in concert. Spatial information alone may not be sufficient for representation, especially in the presence of noise. Additionally the affine transformation is limited in its inability to handle local variation in an object, due to the object's non-rigidity or to the relatively independent movement of its parts, e.g., in human movement.

The optimization problem (2) easily generalizes to integrate multiple invariant features. A representation with multiple features has a spatial component indicating

the location of a feature element. At that location, there may be invariant geometric characteristics, e.g., this point belongs on a curve, or non-geometric invariant features such as color, and texture. Let $X_{jr}$ be the value of feature $r$ associated with point $X_j$. The *location* of point $X_j$ is the null feature. There are $R$ features associated with each point $X_j$ and $Y_k$. Note that the match variable remains the same. The new objective function is identical to the original objective function, (2), appended by the term $\sum_{j,k,r} M_{jk} w_r (X_{jr} - Y_{kr})^2$. The $(X_{jr} - Y_{kr})^2$ quantity captures the similarity between invariant types of features, with $w_r$ a weighting factor for feature $r$. Non-invariant features are not considered. In this way, the point matching algorithm is modified only in the re-initialization of $M(A)$: $M_{jk} = \exp(-\beta(\|X_j - (I + A)Y_k\|^2 + \sum_r w_r (X_{jr} - Y_{kr})^2 - \alpha))$ The rest of the algorithm remains unchanged.

Decomposition of spatial transformations motivates classification of the $B$ individual regions of an object and use of a "quilt" of local affine transformations. In the multiple affine scenario, membership to a region is known on the well-articulated model, but not on the image set. It is assumed that all points that are members of one region undergo the same affine transformation. The model changes by the addition of one subscript to the affine matrix, $A_{\mathbf{b}(k)}$ where $\mathbf{b}(k)$ is an operator that indicates which transformation operates on point $k$. In the algorithm, during the $A(M)$ update, instead of a single update, $B$ updates are done. Denote $K(b) = \{k | \mathbf{b}(k) = b\}$, i.e., all the points that are within region $b$. Then in the affine update, $A_b = A_b^*(M) = (\sum_{j, \, k \in K(b)} M_{jk}(X_j Y_k^T - Y_k Y_k^T))(\sum_{j, \, k \in K(b)} M_{jk} Y_k Y_k^T + \lambda_b I)^{-1}$ However, the theoretical complexity does not change, since the $B$ updates still only require summing over the points.

## 4 Experimental Results: Hand Drawn and Synthetic

The speed for matching point sets of 50 points each is around 20 seconds on an SGI workstation with a R4400 processor. This is true for points in 2D, 3D and with extra features. This can be improved with a tradeoff in accuracy by adopting a looser schedule for the parameter $\beta$ or by changing the stopping criterion.

In the hand drawn examples, the contours of the images are drawn, discretized and then expressed as a set of points in the plane. In Figure (1), the contours of the boy's face were drawn in two different positions, and a subset of the points were extracted to make up the point sets. In each set this was approximately 250 points. Note that even with the change in mood in the two pictures, the corresponding parts of the face are found. However, in Figure (2) spatial information alone is

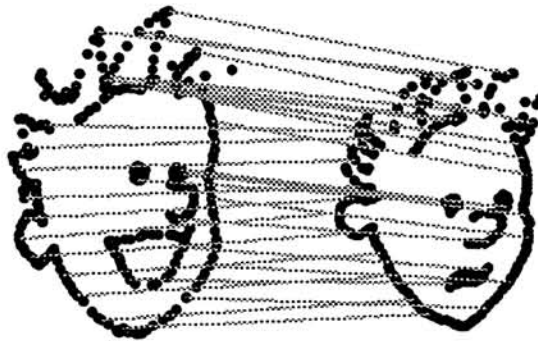

Figure 1: Correspondence with simple point features

insufficient. Although the rotation of the head is not a true affine transformation, it

is a weak perspective projection for which the approximation is valid. Each photo is outlined, generating approximately 225 points in each face. A point on a contour

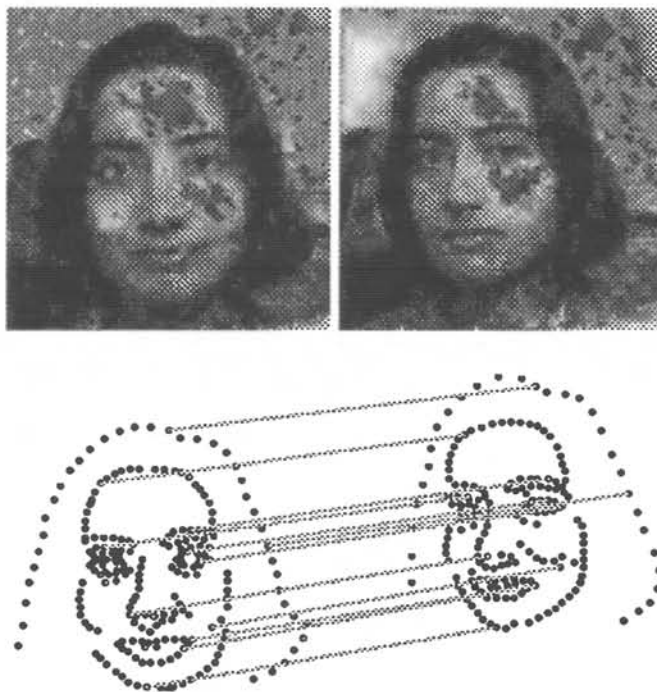

Figure 2: Correspondence with multiple features

has associated with it a feature marker indicating the incident textures. For a human face, we use a binary 4-vector, with a 1 in position $r$ if feature $r$ is present. Specifically, we have used a vector with elements [*skin, hair, lip, eye*]. For example, a point on the line marking the mouth segment the lip from the skin has a feature vector $[1, 0, 1, 0]$. Perceptual organization of the face motivates this type of feature marking scheme. The correspondence is depicted in Figure (2) for a small subset of matches.

Next, we demonstrate how the multiple affine works in recovering the correct correspondence and transformation. The points associated with the standing figure have a marker indicating its part membership. There are six parts in this figure: head, torso, each arm and each leg. The correspondence is shown in Figure (3).

For synthetic data, all 2D and 3D single part experiments used this protocol: The model set was generated uniformly on a unit square. A random affine matrix is generated, whose parameters, $a_{ij}$ are chosen uniformly on a certain interval, which is used to generate the image set. Then, $p_d$ image points are deleted, and Gaussian noise, $N(0, \sigma)$ is added. Finally, spurious points, $p_s$ are added. For the multiple feature scenario, the elements of the feature vector are randomly mislabelled with probability, $P_r$, to represent distortion. For these experiments, 50 model points were generated, and $a_{ij}$ are uniform on an interval of length 1.5. $\sigma \in \{0.01, 0.02, \ldots, 0.08\}$. Point deletions and spurious additions range from 0% to 50% of the image points. The random feature noise associated with non-spatial features has a probability of $P_r = 0.05$. The error measure we use is $e_a = c \sum_{i,j} |a_{ij} - \hat{a}_{ij}|$ where $c = \frac{3}{\# \text{ parameters}} \frac{1}{\text{interval length}}$. $a_{ij}$ and $\hat{a}_{ij}$ are the correct parameter and the computed value, respectively. The constant term $c$ normalizes the measure so that the error equals 1 in the case that the $a_{ij}$ and $\hat{a}_{ij}$ are chosen at random on this interval. The factor 3 in the numerator of this formula follows since

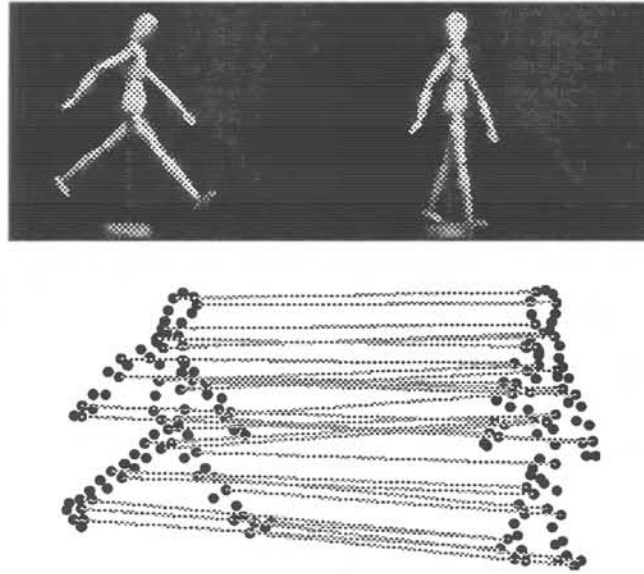

Figure 3: Articulated Matching: Figure with six parts

$E|x - y| = \frac{1}{3}$, when $x$ and $y$ are chosen randomly on the unit interval, and we want to normalize the error. The parameters used in all experiments were: $\beta_{\text{initial}} = .091$, $\beta_{\text{final}} = 100$, $\beta_{\text{update}} = 1.075$, and $T = 4$.

The model has four regions, 24 parameters. Points corresponding to part 1 were centered at $(.5, .5)$, and generated randomly with a diameter of 1.0. For the image set, an affine transformation was applied with a translation diameter of .5, i.e., for $a_{21}, a_{22}$, and the remaining four parameters have a diameter of 1. Points corresponding to regions 2, 3, and 4 were centered at $(-.5, .5), (-.5, -.5), (.5, -.5)$ with model points and transformations generated in a similar fashion. 120 points were generated for the model point set, divided equally among the four parts. Image points were deleted with equal probability from each region. Spurious point were not explicitly added, since the overlapping of parts provides implicit spurious points.

Results for the 2D and 3D (simple point) experiments are in Figure (4). Each data point represents 500 runs for a different randomly generated affine transformation. In all experiments, note that the error for small amounts of noise is approximately equal to that when there is no noise. We performed similar experiments for point sets that are 3-dimensional (12 parameters), but without any feature information. For the experiments with features, shown in Figure (5) we used $R = 4$ features, and $w_r = 0.2, \forall r$. Each data point represents 500 runs. As expected, the inclusion of feature information reduces the error, especially for large $\sigma$. Additionally, Figure (5) details synthetic results for experiments with multiple affines (2D). Each data point represents 70 runs.

## 5    Conclusion

We have developed an affine point matching module, robust in the presence of noise and able to accommodate data of any dimension. The module forms the basis for a non-rigid feature matching scheme in which multiple types of features interact to establish correspondence. Modeling an object in terms of its spatial regions and then using multiple affines to capture local transformations results in a tractable method for non-rigid matching. This non-rigid matching framework arising out of

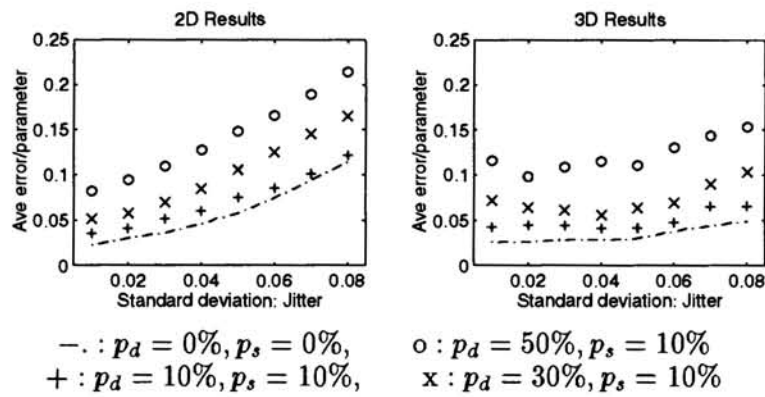

$$-. : p_d = 0\%, p_s = 0\%, \qquad \text{o} : p_d = 50\%, p_s = 10\%$$
$$+ : p_d = 10\%, p_s = 10\%, \qquad \text{x} : p_d = 30\%, p_s = 10\%$$

Figure 4: Synthetic Experiments: 2D and 3D

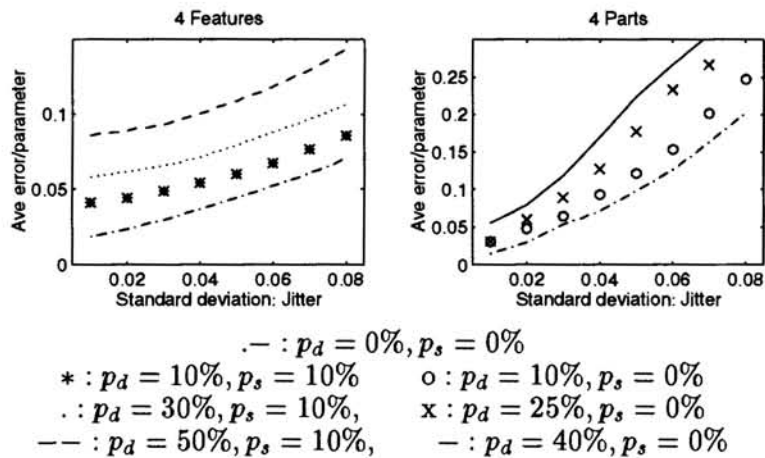

$$.- : p_d = 0\%, p_s = 0\%$$
$$* : p_d = 10\%, p_s = 10\% \qquad \text{o} : p_d = 10\%, p_s = 0\%$$
$$. : p_d = 30\%, p_s = 10\%, \qquad \text{x} : p_d = 25\%, p_s = 0\%$$
$$-- : p_d = 50\%, p_s = 10\%, \qquad - : p_d = 40\%, p_s = 0\%$$

Figure 5: Synthetic Experiments: Multiple features and parts

neural computation is widely applicable in object recognition.

**Acknowledgements:** Our thanks to Eric Mjolsness for many interesting discussions related to the present work.

## Footnotes

[1] e-mail address of authors: lastname-firstname@cs.yale.edu

# References

[1] S. Gold, C. P. Lu, A. Rangarajan, S. Pappu, and E. Mjolsness. New algorithms for 2D and 3D point matching: Pose estimation and correspondence. In G. Tesauro, D. Touretzky, and J. Alspector, editors, *Advances in Neural Information Processing Systems*, volume 7, San Francisco, CA, 1995. Morgan Kaufmann Publishers.

[2] J. Kosowsky and A. Yuille. The invisible hand algorithm: Solving the assignment problem with statistical physics. *Neural Networks*, 7:477–490, 1994.

[3] S. Umeyama. Parameterized point pattern matching and its application to recognition of object families. *IEEE Trans. on Pattern Analysis and Machine Intelligence*, 15:136–144, 1993.

[4] T. Wakahara. Shape matching using LAT and its application to handwritten numeral recognition. *IEEE Trans. in Pattern Analysis and Machine Intelligence*, 16:618–629, 1994.